# Kernel-Based Reinforcement Learning in Average-Cost Problems: An Application to Optimal Portfolio Choice

**Dirk Ormoneit**
Department of Computer Science
Stanford University
Stanford, CA 94305-9010
*ormoneit@cs.stanford.edu*

**Peter Glynn**
EESOR
Stanford University
Stanford, CA 94305-4023

## Abstract

Many approaches to reinforcement learning combine neural networks or other parametric function approximators with a form of temporal-difference learning to estimate the value function of a Markov Decision Process. A significant disadvantage of those procedures is that the resulting learning algorithms are frequently unstable. In this work, we present a new, kernel-based approach to reinforcement learning which overcomes this difficulty and provably converges to a unique solution. By contrast to existing algorithms, our method can also be shown to be consistent in the sense that its costs converge to the optimal costs asymptotically. Our focus is on learning in an average-cost framework and on a practical application to the optimal portfolio choice problem.

## 1  Introduction

Temporal-difference (TD) learning has been applied successfully to many real-world applications that can be formulated as discrete state Markov Decision Processes (MDPs) with unknown transition probabilities. If the state variables are continuous or high-dimensional, the TD learning rule is typically combined with some sort of function approximator – e.g. a linear combination of feature vectors or a neural network – which may well lead to numerical instabilities (see, for example, [BM95, TR96]). Specifically, the algorithm may fail to converge under several circumstances which, in the authors' opinion, is one of the main obstacles to a more wide-spread use of reinforcement learning (RL) in industrial applications. As a remedy, we adopt a non-parametric perspective on reinforcement learning in this work and we suggest a new algorithm that always converges to a unique solution in a finite number of steps. In detail, we assign value function estimates to the states in a sample trajectory and we update these estimates in an iterative procedure. The

updates are based on local averaging using a so-called "weighting kernel". Besides numerical stability, a second crucial advantage of this algorithm is that additional training data always improve the quality of the approximation and eventually lead to optimal performance – that is, our algorithm is consistent in a statistical sense. To the authors' best knowledge, this is the first reinforcement learning algorithm for which consistency has been demonstrated in a continuous space framework. Specifically, the recently advocated "direct" policy search or perturbation methods can by construction at most be optimal in a local sense [SMSM00, VRK00].

Relevant earlier work on local averaging in the context of reinforcement learning includes [Rus97] and [Gor99]. While these papers pursue related ideas, their approaches differ fundamentally from ours in the assumption that the transition probabilities of the MDP are known and can be used for learning. By contrast, kernel-based reinforcement learning only relies on sample trajectories of the MDP and it is therefore much more widely applicable in practice. While our method addresses both discounted- and average-cost problems, we focus on average-costs here and refer the reader interested in discounted-costs to [OS00]. For brevity, we also defer technical details and proofs to an accompanying paper [OG00]. Note that average-cost reinforcement learning has been discussed by several authors (e.g. [TR99]).

The remainder of this work is organized as follows. In Section 2 be provide basic definitions and we describe the kernel-based reinforcement learning algorithm. Section 3 focuses on the practical implementation of the algorithm and on theoretical issues. Sections 4 and 5 present our experimental results and conclusions.

## 2   Kernel-Based Reinforcement Learning

Consider a MDP defined by a sequence of states $X_t$ taking values in $\mathbb{R}^d$, a sequence of actions $a_t$ taking values in $A = \{1, 2, \ldots, M\}$, and a family of transition kernels $\{P_a(x, B) | a \in A\}$ characterizing the conditional probability of the event $X_t \in B$ given $X_{t-1} = x$ and $a_{t-1} = a$. The cost function $c(x, a)$ represents an immediate penalty for applying action $a$ in state $x$. Strategies, policies, or controls are understood as mappings of the form $\mu : \mathbb{R}^d \to A$, and we let $P_{x,\mu}$ denote the probability distribution governing the Markov chain starting from $X_0 = x$ associated with the policy $\mu$. Several regularity conditions are listed in detail in [OG00].

Our goal is to identify policies that are optimal in that they minimize the long-run *average-cost* $\eta_\mu \equiv \lim_{T \to \infty} E_{x,\mu} \left[ \frac{1}{T} \sum_{t=0}^{T-1} c(X_t, \mu(X_t)) \right]$. An optimal policy, $\mu^*$, can be characterized as a solution to the *Average-Cost Optimality Equation (ACOE)*:

$$\eta^* + h^*(x) = \min_a \{c(x, a) + (\Gamma_a h^*)(x)\}, \qquad (1)$$

$$\mu^*(x) = \arg\min_a \{c(x, a) + (\Gamma_a h^*)(x)\}, \qquad (2)$$

where $\eta^*$ is the minimum average-cost and $h^*(x)$ has an interpretation as the *differential value* of starting in $x$ as opposed to drawing a random starting position from the stationary distribution under $\mu^*$. $\Gamma_a$ denotes the conditional expectation operator $(\Gamma_a h)(x) \equiv E_{x,a}[h(X_1)]$, which is assumed to be unknown so that (1) cannot be solved explicitly. Instead, we simulate the MDP using a fixed *proposal strategy* $\bar{\mu}$ in reinforcement learning to generate a sample trajectory as training data. Formally, let $\mathcal{S} \equiv \{z_0, \ldots, z_m\}$ denote such an $m$-step sample trajectory and let

$\mathcal{A} \equiv \{a_0, \ldots, a_{m-1} | a_s = \bar{\mu}(z_s)\}$ and $\mathcal{C} \equiv \{c(z_s, a_s) | 0 \le s < m\}$ be the sequences of actions and costs associated with $\mathcal{S}$. Then our objective can be reformulated as the approximation of $\mu^*$ based on information in $\mathcal{S}$, $\mathcal{A}$, and $\mathcal{C}$. In detail, we will construct an approximate expectation operator, $\hat{\Gamma}_{m,a}$, based on the training data, $\mathcal{S}$, and use this approximation in place of the true operator $\Gamma_a$ in this work. Formally substituting $\hat{\Gamma}_{m,a}$ for $\Gamma_a$ in (1) and (2) gives the *Approximate Average-Cost Optimality Equation (AACOE)*:

$$\hat{\eta}_m + \hat{h}_m(x) = \min_a \{c(x,a) + (\hat{\Gamma}_{m,a}\hat{h}_m)(x)\}, \tag{3}$$

$$\hat{\mu}_m(x) = \arg\min_a \left\{c(x,a) + (\hat{\Gamma}_{m,a}\hat{h}_m)(x)\right\}. \tag{4}$$

Note that, if the solutions $\hat{\eta}_m$ and $\hat{h}_m$ to (3) are well-defined, they can be interpreted as statistical estimates of $\eta^*$ and $h^*$ in equation (1). However, $\hat{\eta}_m$ and $\hat{h}_m$ need not exist unless $\hat{\Gamma}_{m,a}$ is defined appropriately. We therefore employ local averaging in this work to construct $\hat{\Gamma}_{m,a}$ in a way that guarantees the existence of a unique fixed point of (3). For the derivation of the local averaging operator, note that the task of approximating $(\Gamma_a h)(x) = E_{x,a}[h(X_1)]$ can be interpreted alternatively as a regression of the "target" variable $h(X_1)$ onto the "input" $X_0 = x$. So-called kernel-smoothers address regression tasks of this sort by locally averaging the target values in a small neighborhood of $x$. This gives the following approximation:

$$(\hat{\Gamma}_{m,a}h)(x) \equiv \sum_{s=0}^{m-1} k_{m,a}(z_s, x)h(z_{s+1}), \tag{5}$$

$$k_{m,a}(z_s, x) \equiv \frac{\mathbb{1}(a_s = a)\exp\left(-||z_s - x||^2/(2b^2)\right)}{\sum_{s'=0}^{m-1}\mathbb{1}(a_{s'} = a)\exp\left(-||z_{s'} - x||^2/(2b^2)\right)}. \tag{6}$$

In detail, we employ the weighting function or *weighting kernel* $k_{m,a}(z_s, x)$ in (6) to determine the weights that are used for averaging in equation (5). Here $k_{m,a}(z_s, x)$ is a multivariate Gaussian, normalized so as to satisfy the constraints $k_{m,\mu}(z_s, x) > 0$ if $a_s = a$, $k_{m,a}(z_s, x) = 0$ if $a_s \ne a$, and $\sum_{s=0}^{m-1} k_{m,\mu}(z_s, x) = 1$. Intuitively, (5) assesses the future differential cost of applying action $a$ in state $x$ by looking at all times in the training data where $a$ has been applied previously in a state similar to $x$, and by averaging the current differential value estimates at the outcomes of these previous transitions. Because the weights $k_{m,\mu}(z_s, x)$ are related inversely to the distance $||z_s - x||$, transitions originating in the neighborhood of $x$ are most influential in this averaging procedure. A more statistical interpretation of (5) would suggest that ideally we could simply generate a large number of independent samples from the conditional distribution $P_{x,a}$ and estimate $E_{x,a}[h(X_1)]$ using Monte-Carlo approximation. Practically speaking, this approach is clearly infeasible because in order to assess the value of the simulated successor states we would need to sample recursively, thereby incurring exponentially increasing computational complexity. A more realistic alternative is to estimate $\hat{\Gamma}_{m,a}h(x)$ as a local average of the rewards that were generated in previous transitions originating in the neighborhood of $x$, where the membership of an observation $z_s$ in the neighborhood of $x$ is quantified using $k_{m,a}(z_s, x)$. Here the regularization parameter $b$ determines the width of the Gaussian kernel and thereby also the size of the neighborhood used for averaging. Depending on the application, it may be advisable to choose $b$ either fixed or as a location-dependent function of the training data.

## 3 "Self-Approximating Property"

As we illustrated above, kernel-based reinforcement learning formally amounts to substituting the approximate expectation operator $\hat{\Gamma}_{m,a}$ for $\Gamma_a$ and then applying dynamic programming to derive solutions to the approximate optimality equation (3). In this section, we outline a practical implementation of this approach and we present some of our theoretical results. In particular, we consider the relative value iteration algorithm for average-cost MDPs that is described, for example, in [Ber95]. This procedure iterates a variant of equation (1) to generate a sequence of value function estimates, $h_m^k$, that eventually converge to a solution of (1) (or (3), respectively). An important practical problem in continuous state MDPs is that the intermediate functions $h_m^k$ need to be represented explicitly on a computer. This requires some form of function approximation which may be numerically undesirable and computationally burdensome in practice. In the case of kernel-based reinforcement learning, the so-called "self-approximating" property allows for a much more efficient implementation in vector format (see also [Rus97]). Specifically, because our definition of $\hat{\Gamma}_{m,a}h$ in (5) only depends on the values of $h$ at the states in $\mathcal{S}$, the AACOE (3) can be solved in two steps:

$$\hat{\eta}_m + \hat{h}_m(z_s) = \min_a\{c(z_s,a) + (\hat{\Gamma}_{m,a}\hat{h}_m)(z_s)\}, \tag{7}$$

$$\hat{h}_m(x) \equiv \min_a\{c(x,a) + (\hat{\Gamma}_{m,a}\hat{h}_m)(x)\} - \hat{\eta}_m. \tag{8}$$

In other words, we first determine the values of $\hat{h}_m$ at the points in $\mathcal{S}$ using (7) and then compute the values at new locations $x$ in a second step using (8). Note that (7) is a finite equation system by contrast to (3). By introducing the vectors and matrices $\hbar_\mu(i) \equiv h_{m,\mu}(z_i)$, $\mathbf{c}_\mu(i) \equiv c_\mu(z_i)$, $\Phi_\mu(i,j) \equiv k_{m,\mu}(z_j, z_i)$ for $i = 1,\ldots,m$ and $j = 1,\ldots,m$, the relative value iteration algorithm can thus be written conveniently as (for details, see [Ber95, OG00]):

$$\hbar_{new}^k := \min_a(\mathbf{c}_a + \Phi_a\hbar^k), \qquad \hbar^{k+1} := \hbar_{new}^k - \hbar_{new}^k(1). \tag{9}$$

Hence we end up with an algorithm that is analogous to value iteration except that we use the weighting matrix $\Phi_a$ in place of the usual transition probabilities and $\hbar^k$ and $\mathbf{c}_a$ are vectors of points in the training set $\mathcal{S}$ as opposed to vectors of states. Intuitively, (9) assigns value estimates to the states in the sample trajectory and updates these estimates in an iterative fashion. Here the update of each state is based on a local average over the costs and values of the samples in its neighborhood. Since $\Phi_a(i,j) > 0$ and $\sum_{j=1}^m \Phi_a(i,j) = 1$ we can further exploit the analogy between (9) and the usual value iteration in an "artificial" MDP with transition probabilities $\Phi_a$ to prove the following theorem:

**Theorem 1** *The relative value iteration (9) converges to a unique fixed point.*

For details, the reader is referred to [OS00, OG00]. Note that Theorem 1 illustrates a rather unique property of kernel-based reinforcement learning by comparison to alternative approaches. In addition, we can show that – under suitable regularity conditions – kernel-based reinforcement learning is *consistent* in the following sense:

**Theorem 2** *The approximate optimal cost $\hat{\eta}_m$ converges to the true optimal cost $\eta^*$ in the sense that*

$$E_{x_0,\bar{\mu}}|\hat{\eta}_m - \eta^*| \overset{m\to\infty}{\longrightarrow} 0.$$

*Also, the true cost of the approximate strategy $\hat{\mu}_m$ converges to the optimal cost:*

$$E_{x_0,\bar{\mu}}|\eta_{\hat{\mu}_m} - \eta^*| \overset{m \to \infty}{\longrightarrow} 0.$$

Hence $\hat{\mu}_m$ performs as well as $\mu^*$ asymptotically and we can also predict the optimal cost $\eta^*$ using $\hat{\eta}_m$. From a practical standpoint, Theorem 2 asserts that the performance of approximate dynamic programming can be improved by increasing the amount of training data. Note, however, that the computational complexity of approximate dynamic programming depends on the sample size $m$. In detail, the complexity of a single application of (9) is $O(m^2)$ in a naive implementation and $O(m \log m)$ in a more elaborate nearest neighbor approach. This complexity issue prevents the use of very large data sets using the "exact" algorithm described above. As in the case of parametric reinforcement learning, we can of course restrict ourselves to a fixed amount of computational resources simply by discarding observations from the training data or by summarizing clusters of data using "sufficient statistics". Note that the convergence property in Theorem 1 remains unaffected by such an approximation.

## 4  Optimal Portfolio Choice

In this section, we describe the practical application of kernel-based reinforcement learning to an investment problem where an agent in a financial market decides whether to buy or sell stocks depending on the market situation. In the finance and economics literature, this task is known as "optimal portfolio choice" and has created an enormous literature over the past decades. Formally, let $S_t$ symbolize the value of the stock at time $t$ and let the investor choose her portfolio $a_t$ from the set $A \equiv \{0, 0.1, 0.2, \ldots, 1\}$, corresponding to the relative amount of wealth invested in stocks as opposed to an alternative riskless asset. At time $t + 1$, the stock price changes from $S_t$ to $S_{t+1}$, and the portfolio of the investor participates in the price movement depending on her investment choice. Formally, if her wealth at time $t$ is $W_t$, it becomes $W_{t+1} = \left(1 + a_t \frac{S_{t+1} - S_t}{S_t}\right) W_t$ at time $t+1$. To render this simulation as realistic as possible, our investor is assumed to be risk-averse in that her fear of losses dominates her appreciation of gains of equal magnitude. A standard way to express these preferences formally is to aim at maximizing the expectation of a concave "utility function", $\mathcal{U}(z)$, of the final wealth $W_T$. Using the choice $\mathcal{U}(z) = \log(z)$, the investor's utility can be written as $\mathcal{U}(W_T) = \sum_{t=0}^{T-1} \log\left(1 + a_t \frac{S_{t+1} - S_t}{S_t}\right)$. Hence utilities are additive over time, and the objective of maximizing $E[\mathcal{U}(W_T)]$ can be stated in an average-cost framework where $c(x, a) = E_{x,a}\left[\log\left(1 + a \frac{S_{t+1} - S_t}{S_t}\right)\right]$.

We present results using simulated and real stock prices. With regard to the simulated data, we adopt the common assumption in finance literature that stock prices are driven by an Ito process with stochastic, mean-reverting volatility:

$$\begin{aligned} dS_t &= \mu S_t dt + \sqrt{v_t} S_t dB_t, \\ dv_t &= \phi(\bar{v} - v_t)dt + \rho\sqrt{v_t}d\hat{B}_t. \end{aligned}$$

Here $v_t$ is the time-varying volatility, and $B_t$ and $\hat{B}_t$ are independent Brownian motions. The parameters of the model are $\mu = 1.03$, $\bar{v} = 0.3$, $\phi = 10.0$, and $\rho = 5.0$. We

simulated daily data for the period of 13 years using the usual Euler approximation of these equations. The resulting stock prices, volatilities, and returns are shown in Figure 1. Next, we grouped the simulated time series into 10 sets of training and

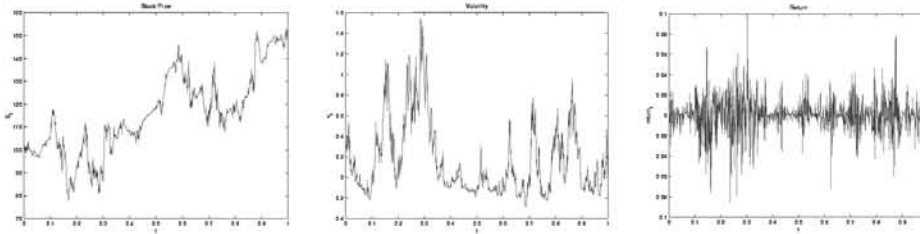

Figure 1: The simulated time-series of stock prices *(left)*, volatility *(middle)*, and daily returns *(right;* $r_t \equiv \log(S_t/S_{t-1})$*)* over a period of one year.

test data such that the last 10 years are used as 10 test sets and the three years preceding each test year are used as training data. Table 1 reports the training and test performances on each of these experiments using kernel-based reinforcement learning and a benchmark buy & hold strategy. Performance is measured using

| Year | Reinforcement Learning | | Buy & Hold | |
|---|---|---|---|---|
| | Training | Test | Training | Test |
| 4 | 0.129753 | 0.096555 | 0.058819 | 0.052533 |
| 5 | 0.125742 | 0.107905 | 0.043107 | 0.081395 |
| 6 | 0.100265 | -0.074588 | 0.053755 | -0.064981 |
| 7 | 0.059405 | 0.201186 | 0.018023 | 0.172968 |
| 8 | 0.082622 | 0.227161 | 0.041410 | 0.197319 |
| 9 | 0.077856 | 0.098172 | 0.074632 | 0.092312 |
| 10 | 0.136525 | 0.199804 | 0.137416 | 0.194993 |
| 11 | 0.145992 | 0.121507 | 0.147065 | 0.118656 |
| 12 | 0.126052 | -0.018110 | 0.125978 | -0.017869 |
| 13 | 0.127900 | -0.022748 | 0.077196 | -0.029886 |

Table 1: Investment performance on the simulated data (initial wealth $W_0 = 100$).

the Sharpe-ratio which is a standard measure of risk-adjusted investment performance. In detail, the Sharpe-ratio is defined as $SR = \log(W_T/W_0)/\tilde{\sigma}$ where $\tilde{\sigma}$ is the standard deviation of $\log(W_t/W_{t-1})$ over time. Note that large values indicate good risk-adjusted performance in years of positive growth, whereas negative values cannot readily be interpreted. We used the root of the volatility (standardized to zero mean and unit variance) as input information and determined a suitable choice for the bandwidth parameter ($b = 1$) experimentally. Our results in Table 1 demonstrate that reinforcement learning dominates buy & hold in eight out of ten years on the training set and in all seven years with positive growth on the test set.

Table 2 shows the results of an experiment where we replaced the artificial time series with eight years of daily German stock index data (DAX index, 1993-2000). We used the years 1996-2000 as test data and the three years preceding each test year for training. As the model input, we computed an approximation of the (root-) volatility using a geometric average of historical returns. Note that the training performance of reinforcement learning always dominates the buy & hold strategy, and the test results are also superior to the benchmark except in the year 2000.

| Year | Reinforcement Learning | | Buy & Hold | |
|---|---|---|---|---|
| | Training | Test | Training | Test |
| 1996 | 0.083925 | 0.173373 | 0.038818 | 0.120107 |
| 1997 | 0.119875 | 0.121583 | 0.119875 | 0.096369 |
| 1998 | 0.123927 | 0.079584 | 0.096183 | 0.035204 |
| 1999 | 0.141242 | 0.094807 | 0.035137 | 0.090541 |
| 2000 | 0.085236 | -0.007878 | 0.081271 | 0.148203 |

Table 2: Investment performance on the DAX data.

## 5 Conclusions

We presented a new, kernel-based reinforcement learning method that overcomes several important shortcomings of temporal-difference learning in continuous-state domains. In particular, we demonstrated that the new approach always converges to a unique approximation of the optimal policy and that the quality of this approximation improves with the amount of training data. Also, we described a financial application where our method consistently outperformed a benchmark model in an artificial and a real market scenario. While the optimal portfolio choice problem is relatively simple, it provides an impressive proof of concept by demonstrating the practical feasibility of our method. Efficient implementations of local averaging for large-scale problems have been discussed in the data mining community. Our work makes these methods applicable to reinforcement learning, which should be valuable to meet the real-time and dimensionality constraints of real-world problems.

**Acknowledgements.** The work of Dirk Ormoneit was partly supported by the Deutsche Forschungsgemeinschaft. Śaunak Sen helped with valuable discussions and suggestions.

## References

[Ber95]   D. P. Bertsekas. *Dynamic Programming and Optimal Control*, volume 1 and 2. Athena Scientific, 1995.

[BM95]   J. A. Boyan and A. W. Moore. Generalization in reinforcement learning: Safely approximating the value function. In *NIPS 7*,1995.

[Gor99]   G. Gordon. *Approximate Solutions to Markov Decision Processes*. PhD thesis, Computer Science Department, Carnegie Mellon University, 1999.

[OG00]   D. Ormoneit and P. Glynn. Kernel-based reinforcement learning in average-cost problems. Working paper, Stanford University. In preparation.

[OS00]   D. Ormoneit and Ś. Sen. Kernel-based reinforcement learning. *Machine Learning*, 2001. To appear.

[Rus97]   J. Rust. Using randomization to break the curse of dimensionality. *Econometrica*, 65(3):487–516, 1997.

[SMSM00]  R. S. Sutton, D. McAllester, S. Singh, and Y. Mansour. Policy gradient methods for reinforcement learning with function approximation. In *NIPS 12*, 2000.

[TR96]   J. N. Tsitsiklis and B. Van Roy. Feature-based methods for large-scale dynamic programming. *Machine Learning*, 22:59–94, 1996.

[TR99]   J. N. Tsitsiklis and B. Van Roy. Average cost temporal-difference learning. *Automatica*, 35(11):1799–1808, 1999.

[VRK00]   J. N. Tsitsiklis V. R. Konda. Actor-critic algorithms. In *NIPS 12*, 2000.